# Location-based Activity Recognition

**Lin Liao, Dieter Fox, and Henry Kautz**
Computer Science & Engineering
University of Washington
Seattle, WA 98195

## Abstract

Learning patterns of human behavior from sensor data is extremely important for high-level activity inference. We show how to extract and label a person's activities and significant places from traces of GPS data. In contrast to existing techniques, our approach simultaneously detects and classifies the significant locations of a person and takes the high-level context into account. Our system uses relational Markov networks to represent the hierarchical activity model that encodes the complex relations among GPS readings, activities and significant places. We apply FFT-based message passing to perform efficient summation over large numbers of nodes in the networks. We present experiments that show significant improvements over existing techniques.

## 1 Introduction

The problem of learning patterns of human behavior from sensor data arises in many areas and applications of computer science, including intelligent environments, surveillance, and assistive technology for the disabled. A focus of recent interest is the use of data from wearable sensors, and in particular, GPS (global positioning system) location data. Such data is used to recognize the high-level activities in which a person is engaged and to determine the relationship between activities and locations that are important to the user [1, 6, 8, 3]. Our goal is to segment the user's day into everyday activities such as "working," "visiting," "travel," and to recognize and label significant locations that are associated with one or more activity, such as "work place," "friend's house," "user's bus stop." Such activity logs can be used, for instance, for automated diaries or long-term health monitoring. Previous approaches to location-based activity recognition suffer from design decisions that limit their accuracy and flexibility:

First, previous work decoupled the subproblem of determining whether or not a geographic location is significant and *should* be assigned a label, from that of labeling places and activities. The first problem was handled by simply assuming that a location is significant if and only if the user spends at least $N$ minutes there, for some fixed threshold $N$ [1, 6, 8, 3]. Some way of restricting the enormous set of all locations recorded for the user to a meaningful subset is clearly necessary. However, in practice, any fixed threshold leads to many errors. Some significant locations, for example, the place where the user drops off his children at school, may be visited only briefly, and so would be excluded by a high threshold. A lower threshold, however, would include too many insignificant locations, for example, a place where the user briefly waited at a traffic light. The inevitable errors cannot

be resolved because information cannot flow from the label assignment process back to the one that determines the domain to be labeled.

Second, concerns for computational efficiency prevented previous approaches from tackling the problem of activity and place labeling in full generality. [1] does not distinguish between places and activities; although [8] does, the implementation limited places to a single activity. Neither approaches model or label the user's activities when moving between places. [6] and [3] learn transportation patterns, but not place labels.

The third problem is one of the underlying causes of the other limitations. The representations and algorithms used in previous work make it difficult to learn and reason with the kinds of *non-local features* that are useful in disambiguating human activity. For a simple example, if a system could learn that a person rarely went to a restaurant more than once a day, then it could correctly give a low probability to an interpretation of a day's data under which the user went to three restaurants. Our previous work [8] used *clique templates* in relational Markov networks for concisely expressing global features, but the MCMC inference algorithm we used made it costly to reason with aggregate features, such as statistics on the number of times a given activity occurs. The ability to *efficiently* leverage global features of the data stream could enhance the scope and accuracy of activity recognition.

This paper presents a unified approach to automated activity and place labeling which overcomes these limitations. Contributions of this work include the following:

- We show how to simultaneously solve the tasks of identifying significant locations and labeling both places and activities from raw GPS data, all in a conditionally trained relational Markov network. Our approach is notable in that nodes representing significant places are dynamically added to the graph during inference. No arbitrary thresholds regarding the time spent at a location or the number of significant places are employed.

- Our model creates a complete interpretation of the log of a user's data, including transportation activities as well as activities performed at particular places. It allows different kinds of activities to be performed at the same location.

- We extend our work on using clique templates for global features to support efficient inference by belief propagation. We introduce, in particular, specialized *Fast Fourier Transform* (FFT) templates for belief propagation over aggregate (counting) features, which reduce computation time by an exponential amount. Although [9] introduced the use of the FFT to compute probability distributions over summations, our work appears to be the first to employ it for full bi-directional belief propagation.

This paper is organized as follows. We begin with a discussion of relational Markov networks and a description of an FFT belief propagation algorithm for aggregate statistical features. Then we explain how to apply RMNs to the problem of location-based activity recognition. Finally, we present experimental results on real-world data that demonstrate significant improvement in coverage and accuracy over previous work.

## 2 Relational Markov Networks and Aggregate Features

### 2.1 Preliminaries

Relational Markov Networks (RMNs) [10] are extensions of Conditional Random Fields (CRFs), which are undirected graphical models that were developed for labeling sequence data [5]. CRFs have been shown to produce excellent results in areas such as natural language processing [5] and computer vision [4]. RMNs extend CRFs by providing a relational language for describing clique structures and enforcing parameter sharing at the template level. Thereby RMNs provide a very flexible and concise framework for defining the features we use in our activity recognition context.

A key concept of RMNs are *relational clique templates*, which specify the structure of a CRF in a concise way. In a nutshell, a clique template $C \in \mathcal{C}$ is similar to a database query (*e.g.*, SQL) in that it selects tuples of nodes from a CRF and connects them into cliques. Each clique template $C$ is additionally associated with a potential function $\phi_C(\mathbf{v}_C)$ that maps values of variables to a non-negative real number. Using a log-linear combination of feature functions, we get $\phi_C(\mathbf{v}_C) = \exp\{\mathbf{w}_C^T \cdot \mathbf{f}_C(\mathbf{v}_C)\}$, where $\mathbf{f}_C()$ defines a feature vector for $C$ and $\mathbf{w}_C^T$ is the transpose of the corresponding weight vector.

An RMN defines a conditional distribution $p(\mathbf{y}|\mathbf{x})$ over labels $\mathbf{y}$ given observations $\mathbf{x}$. To compute such a conditional distribution, the RMN generates a CRF with the cliques specified by the clique templates. All cliques that originate from the same template must share the same weight vector $\mathbf{w}_C$. The resulting cliques factorize the conditional distribution as

$$p(\mathbf{y} \mid \mathbf{x}) = \frac{1}{Z(\mathbf{x})} \prod_{C \in \mathcal{C}} \prod_{\mathbf{v}_C \in C} \exp\{\mathbf{w}_C^T \cdot \mathbf{f}_C(\mathbf{v}_C)\}, \tag{1}$$

where $Z(\mathbf{x})$ is the normalizing partition function.

The weights $\mathbf{w}$ of an RMN can be learned discriminatively by maximizing the log-likelihood of labeled training data [10, 8]. This requires running an inference procedure at each iteration of the optimization and can be very expensive. To overcome this problem, we instead maximize the pseudo-log-likelihood of the training data:

$$L(\mathbf{w}) \equiv \sum_{i=1}^{n} \log p(\mathbf{y}_i \mid \mathrm{MB}(\mathbf{y}_i), \mathbf{w}) - \frac{\mathbf{w}^T \mathbf{w}}{2\sigma^2} \tag{2}$$

where $\mathrm{MB}(\mathbf{y}_i)$ is the Markov Blanket of variable $\mathbf{y}_i$. The rightmost term avoids overfitting by imposing a zero-mean, Gaussian shrinkage prior on each component of the weights [10]. In the context of place labeling, [8] showed how to use non-zero mean priors in order to transfer weights learned for one person to another person. In our experiments, learning the weights using pseudo-log-likelihood is very efficient and performs well in our tests.

In our previous work [8] we used MCMC for inference. While this approach performed well for the models considered in [8], it does not scale to more complex activity models such as the one described here. Taskar and colleagues [10] relied on belief propagation (BP) for inference. The BP (sum-product) algorithm converts a CRF to a pairwise representation and performs message passing, where the message from node $i$ to its neighbor $j$ is computed as

$$m_{ij}(\mathbf{y}_j) = \sum_{\mathbf{y}_i} \phi(\mathbf{y}_i)\phi(\mathbf{y}_i, \mathbf{y}_j) \prod_{k \in n(i) \setminus j} m_{ki}(\mathbf{y}_i), \tag{3}$$

where $\phi(\mathbf{y}_i)$ is a local potential, $\phi(\mathbf{y}_i, \mathbf{y}_j)$ is a pairwise potential, and $\{n(i) \setminus j\}$ denotes $i$'s neighbors other than $j$. All messages are updated iteratively until they (possibly) converge. However, our model takes into account aggregate features, such as summation. Performing aggregation would require the generation of cliques that contain all nodes over which the aggregation is performed. Since the complexity of standard BP is exponential in the number of nodes in the largest clique, aggregation can easily make BP intractable.

## 2.2 Efficient summation templates

In our model, we address the inference of aggregate cliques at the template level within the framework of BP. Each type of aggregation function is associated with a *computation template* that specifies how to propagate messages through the clique. In this section, we discuss an efficient computation template for summation.

To handle summation cliques with potentially large numbers of addends, our summation template dynamically builds a *summation tree*, which is a pairwise Markov network as shown in Fig. 1(a). In a summation tree, the leaves are the original addends and each

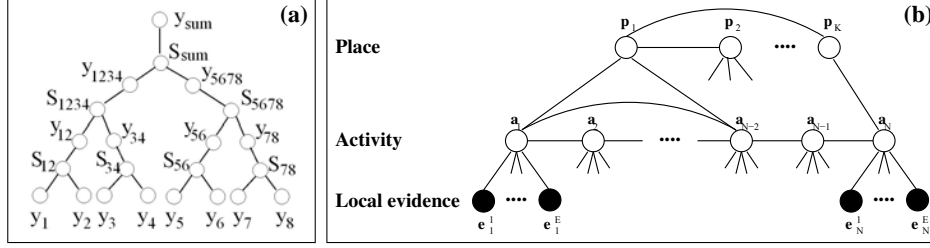

Figure 1: (a) Summation tree that represents $\mathbf{y}_{sum} = \sum_{i=1}^{8} \mathbf{y}_i$, where the $S_i$'s are auxiliary nodes to ensure the summation relation. (b) CRF for labeling activities and places. Each activity node $a_i$ is connected to $E$ observed local evidence nodes $e_i^1$ to $e_i^E$. Place nodes $p_i$ are generated based on the inferred activities and each place is connected to all activity nodes that are within a certain distance.

internal node $\mathbf{y}_{jk}$ represents the sum of its two children $\mathbf{y}_j$ and $\mathbf{y}_k$, and this sum relation is encoded by an auxiliary node $S_{jk}$ and its potential. The state space of $S_{jk}$ consists of the joint (cross-product) state of its neighbors $\mathbf{y}_j$, $\mathbf{y}_k$, and $\mathbf{y}_{jk}$. It is easy to see that the summation tree guarantees that the root represents $\mathbf{y}_{sum} = \sum_{i=1}^{n} \mathbf{y}_i$, where $\mathbf{y}_1$ to $\mathbf{y}_n$ are the leaves of the tree. To define the BP protocol for summation trees, we need to specify two types of messages: an *upward message* from an auxiliary node to its parent (*e.g.*, $m_{S_{12}\mathbf{y}_{12}}$), and a *downward message* from an auxiliary node to one of its two children (*e.g.*, $m_{S_{12}\mathbf{y}_1}$).

**Upward message update:** Starting with Equation (3), we can update an upward message $m_{S_{ij}\mathbf{y}_{ij}}$ as follows.

$$m_{S_{ij}\mathbf{y}_{ij}}(\mathbf{y}_{ij}) = \sum_{\mathbf{y}_i,\mathbf{y}_j} \phi_S(\mathbf{y}_i,\mathbf{y}_j,\mathbf{y}_{ij})\, m_{\mathbf{y}_i S_{ij}}(\mathbf{y}_i)\, m_{\mathbf{y}_j S_{ij}}(\mathbf{y}_j)$$

$$= \sum_{\mathbf{y}_i} m_{\mathbf{y}_i S_{ij}}(\mathbf{y}_i)\, m_{\mathbf{y}_j S_{ij}}(\mathbf{y}_{ij} - \mathbf{y}_i) \tag{4}$$

$$= \mathcal{F}^{-1}\left(\mathcal{F}(m_{\mathbf{y}_i S_{ij}}(\mathbf{y}_i)) \cdot \mathcal{F}(m_{\mathbf{y}_j S_{ij}}(\mathbf{y}_j))\right) \tag{5}$$

where $\phi_S(\mathbf{y}_i,\mathbf{y}_j,\mathbf{y}_{ij})$ is the local potential of $S_{ij}$ encoding the equality $\mathbf{y}_{ij} = \mathbf{y}_i + \mathbf{y}_j$. (4) follows because all terms not satisfying the equality disappear. Therefore, message $m_{S_{ij}\mathbf{y}_{ij}}$ is the *convolution* of $m_{\mathbf{y}_i S_{ij}}$ and $m_{\mathbf{y}_j S_{ij}}$. (5) follows from the *convolution theorem*, which states that the Fourier transform of a convolution is the point-wise product of Fourier transforms [2], where $\mathcal{F}$ and $\mathcal{F}^{-1}$ represent the Fourier transform and its inverse, respectively. When the messages are discrete functions, the Fourier transform and its inverse can be computed efficiently using the Fast Fourier Transform (FFT) [2, 9]. The computational complexity of one summation using FFT is $O(k \log k)$, where $k$ is the maximum number of states in $\mathbf{y}_i$ and $\mathbf{y}_j$.

**Downward message update:** We also allow messages to pass from sum variables downward to its children. This is necessary if we want to use the belief on sum variables (*e.g.*, knowledge on the number of homes) to change the distribution of individual variables (*e.g.*, place labels). From Equation (3) we get the downward message $m_{S_{ij}\mathbf{y}_i}$ as

$$m_{S_{ij}\mathbf{y}_i}(\mathbf{y}_i) = \sum_{\mathbf{y}_j,\mathbf{y}_{ij}} \phi_S(\mathbf{y}_i,\mathbf{y}_j,\mathbf{y}_{ij}) m_{\mathbf{y}_j S_{ij}}(\mathbf{y}_j) m_{\mathbf{y}_{ij} S_{ij}}(\mathbf{y}_{ij})$$

$$= \sum_{\mathbf{y}_j} m_{\mathbf{y}_j S_{ij}}(\mathbf{y}_j) m_{\mathbf{y}_{ij} S_{ij}}(\mathbf{y}_i + \mathbf{y}_j) \tag{6}$$

$$= \mathcal{F}^{-1}\left(\overline{\mathcal{F}}(m_{\mathbf{y}_j S_{ij}}(\mathbf{y}_j)) \cdot \mathcal{F}(m_{\mathbf{y}_{ij} S_{ij}}(\mathbf{y}_{ij}))\right) \tag{7}$$

where (6) again follows from the sum relation. Note that the downward message $m_{S_{ij}\mathbf{y}_i}$ turns out to be the *correlation* of messages $m_{\mathbf{y}_j S_{ij}}$ and $m_{\mathbf{y}_{ij} S_{ij}}$. (7) follows from the *correlation theorem* [2], which is similar to the convolution theorem except, for correlation, we must compute the *complex conjugate* of the first Fourier transform, denoted as $\overline{\mathcal{F}}$. Again, for discrete messages, (7) can be evaluated efficiently using FFT.

At each level of a summation tree, the number of messages (nodes) is reduced by half and the size of each message is doubled. Suppose the tree has $n$ upward messages at the bottom and the maximum size of a message is $k$. For large summation trees where $n \gg k$, the total complexity of updating the upward messages at all the $\log n$ levels follows now as

$$\sum_{i=1}^{\log n} \frac{n}{2^i} \cdot O\left(2^{i-1} k \log 2^{i-1} k\right) = O\left(\frac{n}{2} \sum_{i=1}^{\log n} \log 2^{i-1}\right) = O(n \log^2 n) \qquad (8)$$

Similar reasoning shows that the complexity of the downward pass is $O(n \log^2 n)$ as well. Therefore, updating all messages in a summation clique takes $O(n \log^2 n)$ instead of time exponential in $n$, as would be the case for a non-specialized implementation of aggregation.

## 3  Location-based Activity Model

### 3.1  Overview

To recognize activities and places, we first segment raw GPS traces by grouping consecutive GPS readings based on their spatial relationship. This segmentation can be performed by simply combining all consecutive readings that are within a certain distance from each other (10m in our implementation). However, it might be desirable to associate GPS traces to a street map, for example, in order to relate locations to addresses in the map. To jointly estimate the GPS to street association and trace segmentation, we construct an RMN that takes into account the spatial relationship and temporal consistency between the measurements and their associations (see [7] for more details). In this section, we focus on inferring activities and types of significant places after segmentation. To do so, we construct a hierarchical RMN that explicitly encodes the relations between activities and places. A CRF instantiated from the RMN is shown in Fig. 1(b). At the lower level of the hierarchy, each activity node is connected to various features, summarizing information resulting from the GPS segmentation. These features include:

- Temporal information such as time of day, day of week, and duration of the stay;
- Average speed through a segment, for discriminating transportation modes;
- Information extracted from geographic databases, such as whether a location is close to a bus route or bus stop, and whether it is near a restaurant or store;
- Additionally, each activity node is connected to its neighbors. These features measure compatibility between types of activities at neighboring nodes in the trace.

Our model also aims at determining those places that play a significant role in the activities of a person, such as home, work place, friend's home, grocery stores, restaurants, and bus stops. Such *significant places* comprise the upper level of the CRF shown in Fig. 1(b). However, since these places are not known a priori, we must additionally *detect* a person's significant places. To incorporate place detection into our system, we use an iterative algorithm that re-estimates activities and places. Before we describe this algorithm, let us first look at the features that are used to determine the types of significant places under the assumption that the locations and number of these places are known.

- The activities that occur at a place strongly indicate the type of the place. For example, at a friends' home people either visit or pick up / drop off someone. Our features consider the *frequencies* of the different activities at a place. This is done by generating a clique for each place that contains all activity nodes in its vicinity. For example, the nodes $p_1$, $a_1$, and $a_{N-2}$ in Fig. 1(b) form such a clique.
- A person usually has only a limited number of different homes or work places. We add two additional summation cliques that count the number of homes and work places. These counts provide soft constraints that bias the system to generate interpretations with reasonable numbers of homes and work places.

1.  **Input:** GPS trace $\langle g_1, g_2, \ldots, g_T \rangle$ and iteration counter $i := 0$
2.  $\left( \langle a_1, \ldots, a_N \rangle, \langle e_1^1, \ldots, e_1^E, \ldots \rangle \right) :=$ trace_segmentation $(\langle g_1, g_2, \ldots, g_T \rangle)$
3.  *// Generate CRF containing activity and local evidence (lower two levels in Fig. 1(b))*
    $\text{CRF}_0 :=$ instantiate_crf $\left( \langle \, \rangle, \langle a_1, \ldots, a_N \rangle, \langle e_1^1, \ldots, e_1^E, \ldots \rangle \right)$
4.  $\mathbf{a}^*_0 :=$ BP_inference( $\text{CRF}_0$) *// infer sequence of activities*
5.  **do**
6.  $\quad$ i := i + 1
7.  $\quad \langle p_1, \ldots, p_K \rangle_i :=$ generate_places($\mathbf{a}^*_{i-1}$) *// Instantiate places*
8.  $\quad \text{CRF}_i :=$ instantiate_crf $\left( \langle p_1, \ldots, p_K \rangle_i, \langle a_1, \ldots, a_N \rangle, \langle e_1^1, \ldots, e_1^E, \ldots \rangle \right)$
9.  $\quad \langle \mathbf{a}_i^*, \mathbf{p}_i^* \rangle :=$ BP_inference( $\text{CRF}_i$) *// inference in complete CRF*
10. **until** $\mathbf{a}_i^* = \mathbf{a}_{i-1}^*$
11. **return** $\langle \mathbf{a}_i^*, \mathbf{p}_i^* \rangle$

Table 1: Algorithm for extracting and labeling activities and significant places.

Note that the above two types of aggregation features can generate large cliques in the CRF, which could make standard inference intractable. In our inference, we use the optimized summation templates discussed in Section 2.2.

### 3.2 Place Detection and Labeling Algorithm

Table 1 summarizes our algorithm for efficiently constructing a CRF that jointly estimates a person's activities and the types of his significant places. The algorithm takes as input a GPS trace. In Step 2 and 3, this trace is segmented into activities $a_i$ and their local evidence $e_i^j$, which are then used to generate $\text{CRF}_0$ *without significant places*. BP inference is first performed in this CRF so as to determine the activity estimate $\mathbf{a}^*_0$, which consists of a sequence of locations and the most likely activity performed at that location (Step 4). Within each iteration of the loop starting at Step 5, such an activity estimate is used to extract a set of significant places. This is done by classifying individual activities in the sequence according to whether or not they belong to a significant place. For instance, while walking, driving a car, or riding a bus are not associated with significant places, working or getting on or off the bus indicate a significant place. All instances at which a *significant activity* occurs generate a place node. Because a place can be visited multiple times within a sequence, we perform clustering and merge duplicate places into the same place node. This classification and clustering is performed by the algorithm generate_places() in Step 7. These places are added to the model and BP is performed in this complete CRF. Since a $\text{CRF}_i$ can have a different structure than the previous $\text{CRF}_{i-1}$, it might generate a different activity sequence. If this is the case, the algorithm returns to Step 5 and re-generates the set of places using the improved activity sequence. This process is repeated until the activity sequence does not change. In our experiments we observed that this algorithm converges very quickly, typically after three or four iterations.

## 4 Experimental Results

In our experiments, we collected GPS data traces from four different persons, approximately seven days of data per person. The data from each person consisted of roughly 40,000 GPS measurements, resulting in about 10,000 10m segments. We used leave-one-out cross-validation for evaluation. Learning from three persons' data took about one minute and BP inference on the last person's data converged within one minute.

### Extracting significant places

We compare our model with a widely-used approach that uses a time threshold to determine whether or not a location is significant [1, 6, 8, 3]. We use four different thresholds from

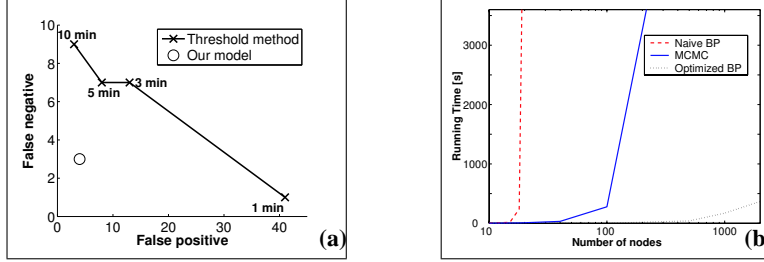

Figure 2: (a) Accuracy of extracting places. (b) Computation times for summation cliques.

| Truth | Inferred labels | | | | | | | FN |
|---|---|---|---|---|---|---|---|---|
| | Work | Sleep | Leisure | Visit | Pickup | On/off car | Other | |
| Work | 12 / 11 | 0 | 0 / 1 | 0 | 0 | 0 | 1 | 0 |
| Sleep | 0 | 21 | 1 | 2 | 0 | 0 | 0 | 0 |
| Leisure | 2 | 0 | 20 / 17 | 1 / 4 | 0 | 0 | 3 | 0 |
| Visiting | 0 | 0 | 0 / 2 | 7 / 5 | 0 | 0 | 2 | 0 |
| Pickup | 0 | 0 | 0 | 0 | 1 | 0 | 0 | 2 |
| On/Off car | 0 | 0 | 0 | 0 | 1 | 13 / 12 | 0 | 2 / 3 |
| Other | 0 | 0 | 0 | 0 | 0 | 0 | 37 | 1 |
| FP | 0 | 0 | 0 | 0 | 2 | 2 | 3 | - |

Table 2: Activity confusion matrix of cross-validation data with (left values) and without (right values) considering places for activity inference (FN and FP are false negatives and false positives).

1 minute to 10 minutes, and we measure the false positive and false negative locations extracted from the GPS traces. As shown in Fig. 2(a), any fixed threshold is not satisfactory: low thresholds have many false positives, and high thresholds result in many false negatives. In contrast, our model performs much better: it only generates 4 false positives and 3 false negative. This experiment shows that using high-level context information drastically improves the extraction of significant places.

**Labeling places and activities**

In our system the labels of activities generate instances of places, which then help to better estimate the activities occurring in their spatial area. The confusion matrix given in Table 2 summarizes the activity estimation results achieved with our system on the cross-validation data. The results are given with and without taking the detected places into account. More specifically, without places are results achieved by $CRF_0$ generated by Step 4 of the algorithm in Table 1, and results with places are those achieved after model convergence. When the results of both approaches are identical, only one number is given, otherwise, the first number gives the result achieved with the complete model. The table shows two main results. First, the accuracy of our approach is quite high, especially when considering that the system was evaluated on only one week of data and was trained on only three weeks of data collected by different persons. Second, performing joint inference over activities and places increases the quality of inference. The reason for this is that a place node connects all the activities occurring in its spatial area so that these activities can be labeled in a more consistent way. A further evaluation of the detected places showed that our system achieved 90.6% accuracy in place detection and labeling (see [7] for more results).

**Efficiency of inference**

We compared our optimized BP algorithm using FFT summation cliques with inference based on MCMC and regular BP, using the model and data from [8]. Note that a naive implementation of BP is exponential in the number of nodes in a clique. In our experiments, the test accuracies resulting from using the different algorithms are almost identical. Therefore, we only focus on comparing the efficiency and scalability of summation aggregations. The running times for the different algorithms are shown in Fig. 2(b). As can be seen, naive

BP becomes extremely slow for only 20 nodes, MCMC only works for up to 500 nodes, while our algorithm can perform summation for 2,000 variables within a few minutes.

## 5 Conclusions

We provided a novel approach to performing location-based activity recognition. In contrast to existing techniques, our approach uses one consistent framework for both low-level inference and the extraction of a person's significant places. Thereby, our model is able to take high-level context into account in order to detect the significant locations of a person. Furthermore, once these locations are determined, they help to better detect low-level activities occurring in their vicinity.

Summation cliques are extremely important to introduce long-term, soft constraints into activity recognition. We show how to incorporate such cliques into belief propagation using bi-directional FFT computations. The clique templates of RMNs are well suited to specify such clique-specific inference mechanisms and we are developing additional techniques, including clique-specific MCMC and local dynamic programming.

Our experiments based on traces of GPS data show that our system significantly outperforms existing approaches. We demonstrate that the model can be trained from a group of persons and then applied successfully to a different person, achieving more than 85% accuracy in determining low-level activities and above 90% accuracy in detecting and labeling significant places. In future work, we will add more sensor data, including accelerometers, audio signals, and barometric pressure. Using the additional information provided by these sensors, we will be able to perform more fine-grained activity recognition.

### Acknowledgments
The authors would like to thank Jeff Bilmes for useful comments. This work has partly been supported by DARPA's ASSIST and CALO Programme (contract numbers: NBCH-C-05-0137, SRI subcontract 27-000968) and by the NSF under grant number IIS-0093406.

## References

[1] D. Ashbrook and T. Starner. Using GPS to learn significant locations and predict movement across multiple users. *Personal and Ubiquitous Computing*, 7(5), 2003.

[2] E. Oran Brigham. *Fast Fourier Transform and Its Applications*. Prentice Hall, 1988.

[3] V. Gogate, R. Dechter, C. Rindt, and J. Marca. Modeling transportation routines using hybrid dynamic mixed networks. In *Proc. of the Conference on Uncertainty in Artificial Intelligence*, 2005.

[4] S. Kumar and M. Hebert. Discriminative random fields: A discriminative framework for contextual interaction in classification. In *Proc. of the International Conference on Computer Vision*, 2003.

[5] J. Lafferty, A. McCallum, and F. Pereira. Conditional random fields: Probabilistic models for segmenting and labeling sequence data. In *Proc. of the International Conference on Machine Learning*, 2001.

[6] L. Liao, D. Fox, and H. Kautz. Learning and inferring transportation routines. In *Proc. of the National Conference on Artificial Intelligence*, 2004.

[7] L. Liao, D. Fox, and H. Kautz. Hierarchical conditional random fields for GPS-based activity recognition. In *Proc. of the 12th International Symposium of Robotics Research (ISRR)*, 2005.

[8] L. Liao, D. Fox, and H. Kautz. Location-based activity recognition using relational Markov networks. In *Proc. of the International Joint Conference on Artificial Intelligence*, 2005.

[9] Yongyi Mao, Frank R. Kschischang, and Brendan J. Frey. Convolutional factor graphs as probabilistic models. In *Proc. of the Conference on Uncertainty in Artificial Intelligence*, 2004.

[10] B. Taskar, P. Abbeel, and D. Koller. Discriminative probabilistic models for relational data. In *Proc. of the Conference on Uncertainty in Artificial Intelligence*, 2002.
